# A Reinforcement Learning Variant for Control Scheduling

Aloke Guha
Honeywell Sensor and System Development Center
3660 Technology Drive
Minneapolis, MN 55417

## Abstract

We present an algorithm based on reinforcement and state recurrence learning techniques to solve control scheduling problems. In particular, we have devised a simple learning scheme called "handicapped learning", in which the weights of the associative search element are reinforced, either positively or negatively, such that the system is forced to move towards the desired setpoint in the shortest possible trajectory. To improve the learning rate, a variable reinforcement scheme is employed: negative reinforcement values are varied depending on whether the failure occurs in handicapped or normal mode of operation. Furthermore, to realize a simulated annealing scheme for accelerated learning, if the system visits the same failed state successively, the negative reinforcement value is increased. In examples studied, these learning schemes have demonstrated high learning rates, and therefore may prove useful for in-situ learning.

## 1 INTRODUCTION

Reinforcement learning techniques have been applied successfully for simple control problems, such as the pole-cart problem [Barto 83, Michie 68, Rosen 88] where the goal was to maintain the pole in a quasistable region, but not at specific setpoints. However, a large class of continuous control problems require maintaining the system at a desired operating point, or setpoint, at a given time. We refer to this problem as the basic setpoint control problem [Guha 90], and have shown that reinforcement learning can be used, not surprisingly, quite well for such control tasks. A more general version of the same problem requires steering the system from some

initial or starting state to a desired state or setpoint at specific times without knowledge of the dynamics of the system. We therefore wish to examine how control scheduling tasks, where the system must be steered through a sequence of setpoints at specific times, can be learned. Solving such a control problem without explicit modeling of the system or plant can prove to be beneficial in many adaptive control tasks.

To address the control scheduling problem, we have derived a learning algorithm called handicapped learning. Handicapped learning uses a nonlinear encoding of the state of the system, a new associative reinforcement learning algorithm, and a novel reinforcement scheme to explore the control space to meet the scheduling constraints. The goal of handicapped learning is to learn the control law necessary to steer the system from one setpoint to another. We provide a description of the state encoding and associative learning in Section 2, the reinforcement scheme in Section 3, the experimental results in Section 4, and the conclusions in Section 5.

## 2 REINFORCEMENT LEARNING STRATEGY: HANDICAPPED LEARNING

Our earlier work on regulatory control using reinforcement learning [Guha 90] used a simple linear coded state representation of the system. However, when considering multiple setpoints in a schedule, a linear coding of high-resolution results in a combinatorial explosion of states. To avoid this curse of dimensionality, we have adopted a simple nonlinear encoding of the state space. We describe this first.

### 2.1 STATE ENCODING

To define the states in which reinforcement must be provided to the controller, we set tolerance limits around the desired setpoint, say $X_d$. If the tolerance of operation defined by the level of control sophistication required in the problem is T, then the controller is defined to fail if $|X(t) - X_d| > T$ as described in our earlier work in [Guha 90].

The controller must learn to maintain the system within this tolerance window. If the range, R, of possible values of the setpoint or control variable $X(t)$ is significantly greater than the tolerance window, then the number of states required to define the setpoint will be large. We therefore use a nonlinear coding of the control variable. Thus, if the level of discrimination within the tolerance window is $2T/n$, then the number of states required to represent the control variable is $(n + 2)$ where the two added states represent the states, $(X(t) - X_d) > T$ and $(X(t) - X_d) < -T$. With this representation scheme, any continuous range of setpoints can be represented with very high resolution but without the explosion in state space.

The above state encoding will be used in our associative reinforcement learning algorithm, handicapped learning, which we describe next.

## 2.2 HANDICAPPED LEARNING ALGORITHM

Our reinforcement learning strategy is derived from the Associative Search Element/Adaptive Heuristic Critic (ASE/AHC) algorithm [Barto 83, Anderson 86]. We have considered a binary control output, $y(t)$:

$$y(t) = f(\sum_i w_i(t)x_i(t) + noise(t)) \qquad (1)$$

where f is the thresholding step function, and $x_i(t)$, $0 \leq i \leq N$, is the current decoded state, that is, $x_i(t) = 1$ when the system is in the ith state and 0 otherwise. As in ASE, the added term noise(t) facilitates stochastic learning. Note that the learning algorithm can be easily extended to continuous valued outputs, the nature of the continuity is determined by the thresholding function.

We incorporate two learning heuristics: state recurrence [Rosen 88] and a newly introduced heuristic called "handicapped learning". The controller is in the handicapped learning mode if a flag, H, is set high. H is defined as follows:

$$H = 0, \quad if \ |X(t) - X_d| < T$$
$$= 1, \quad otherwise \qquad (2)$$

The handicap mode provides a mechanism to modify the reinforcement scheme. In this mode the controller is allowed to explore the search space of action sequences, to steer to a new setpoint, without "punishment" (negative reinforcement). The mode is invoked when the system is at a valid setpoint $X_1(t_1)$ at time $t_1$, but must be steered to the new setpoint $X_2$ outside the tolerance window, that is, $|X_1 - X_2| > T$, at time $t_2$. Since both setpoints are valid operating points, these setpoints as well as all points within the possible optimal trajectories from $X_1$ to $X_2$ cannot be deemed to be failure states. Further, by following a special reinforcement scheme during the handicapped mode, one can enable learning and facilitate the controller to find the optimal trajectory to steer the system from one setpoint to another.

The weight updating rule used during setpoint schedule learning is given by equation (3):

$$w_i(t+1) = w_i(t) + \alpha \ r_1(t) \ e_i(t) + \alpha_2 \ r_2(t) \ e_{2i}(t) + \alpha_3 \ r_3(t) \ e_{3i}(t) \qquad (3)$$

where the term $\alpha \ r_1(t) \ e_i(t)$ is the basic associative learning component, $r_1(t)$ the heuristic reinforcement, and $e_i(t)$ the eligibility trace of the state $x_i(t)$ [Barto 83].

The third term in equation (3) is the state recurrence component for reinforcing short cycles [Rosen 88]. Here $\alpha_2$ is a constant gain, $r_2(t)$ is a positive constant reward, and $e_{2i}$ the state recurrence eligibility is defined as follows:

$$e_{2i}(t) = \beta_2 \ x_i(t)y(t_{i,last})/(\beta_2 + t - t_{i,last}), \qquad if \ (t - t_{i,last}) > 1 \ and \ H = 0$$
$$= 0, \quad otherwise \qquad (4)$$

where $\beta_2$ is a positive constant, and $t_{i,last}$ is the last time the system visited the ith state. The eligibility function in equation (4) reinforces shorter cycles more than longer cycles, and improve control when the system is within a tolerance window.

The fourth term in equation (3) is the handicapped learning component. Here $\alpha_3$ is a constant gain, $r_3(t)$ is a positive constant reward and $e_{3i}$ the handicapped learning eligibility is defined as follows:

$$
\begin{aligned}
e_{3i}(t) &= -\beta_3\, x_i(t)y(t_{i,last})/(\beta_3 + t - t_{i,last}), &\quad \text{if } H = 1 \\
&= 0, \quad \text{otherwise}
\end{aligned}
\tag{5}
$$

where $\beta_3$ is a positive constant. While state recurrence promotes short cycles around a desired operating point, handicapped learning forces the controller to move away from the current operating point $X(t)$. The system enters the handicapped mode whenever it is outside the tolerance window around the desired setpoint. If the initial operating point $X_i$ $(= X(0))$ is outside the tolerance window of the desired setpoint $X_d$, $|X_i - X_d| > T$, the basic AHC network will always register a failure. This failure situation is avoided by invoking the handicapped learning described above. By setting absolute upper and lower limits to operating point values, the controller based on handicapped learning can learn the correct sequence of actions necessary to steer the system to the desired operating point $X_d$.

The weight update equations for the critic in the AHC are unchanged from the original AHC and we do not list them here.

## 3 REINFORCEMENT SCHEMES

Unlike in previous experiments by other researchers, we have constructed the reinforcement values used during learning to be multivalued, and not binary. The reinforcement to the critic is negative–both positive and negative reinforcements are used. There are two forms of failure that can occur during setpoint control. First, the controller can reach the absolute upper or lower limits. Second, there may be a timeout failure in the handicapped mode. By design, when the controller is in handicapped mode, it is allowed to remain there for only $T_L$, determined by the average control step $A_y$ and the error between the current operating point and the desired setpoint:

$$
T_L = k\, A_y\, (X_0 - X_d)
\tag{6}
$$

where $X_0$ is the initial setpoint, and $k$ some constant. The negative reinforcement provided to the controller is higher if the absolute limits of the operating point is reached.

We have implemented a more interesting reinforcement scheme that is somewhat similar to simulated annealing. If the system fails in the same state on two successive trials, the negative reinforcement is increased. The primary reinforcement function can be defined as follows:

$$r_j(k + 1) = r_i(k) - r0, \qquad \text{if } i = j$$
$$= r1, \qquad \text{if } i \neq j \qquad\qquad (7)$$

where $r_i(k)$ is the negative reinforcement provided if the system failed in state i during trial k, and r0 and r1 are constants.

## 4  EXPERIMENTS AND RESULTS

Two different setpoint control experiments have been conducted. The first was the basic setpoint control of a continuous stirred tank reactor in which the temperature must be held at a desired setpoint. That experiment successfully demonstrated the use of reinforcement learning for setpoint control of a highly nonlinear and unstable process [Guha 90]. The second recent experiment has been on evaluating the handicapped learning strategy for an environmental controller where the controller must learn to control the heating system to maintain the ambient temperature specified by a time-temperature schedule. Thus, as the external temperature varies, the network must adapt the heating (ON) and (OFF) control sequence so as to maintain the environment at the desired temperature as quickly as possible. The state information describing system is composed of the time interval of the schedule, the current heating state (ON/OFF), and the error or the difference between desired and current ambient or interior temperature. The heating and cooling rates are variable: the heating rate decreases while the cooling rate increases exponentially as the exterior temperature falls below the ambient or controlled temperature.

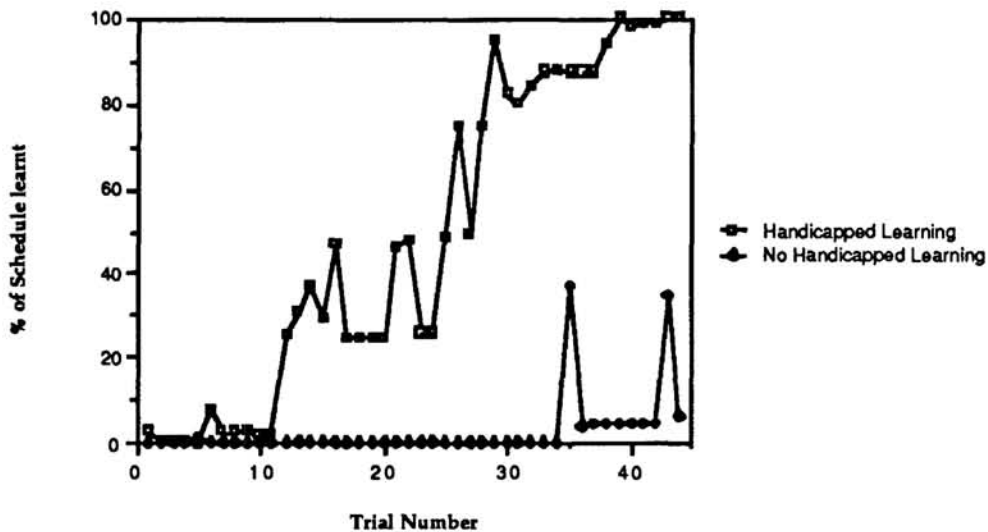

Figure 1: Rate of Learning with and without Handicapped Learning

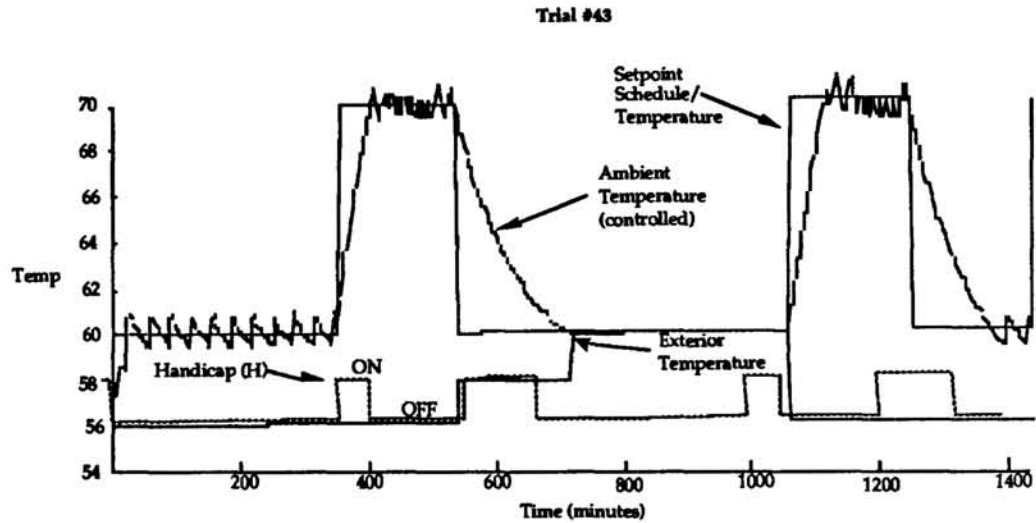

Figure 2: Time-Temperature Plot of Controlled Environment at Forty-third Trial

The experiments on the environmental controller consisted of embedding a daily setpoint schedule that contains six setpoints at six specific times. Trails were conducted to train the controller. Each trial starts at the beginning of the schedule (time = 0). The setpoints typically varied in the range of 55 to 75 degrees. The desired tolerance window was 1 degree. The upper and lower limits of the controlled temperature were set arbitrarily at 50 and 80 degrees, respectively. Control actions were taken every 5 minutes. Learning was monitored by examining how much of the schedule was learnt correctly as the number of trials increased.

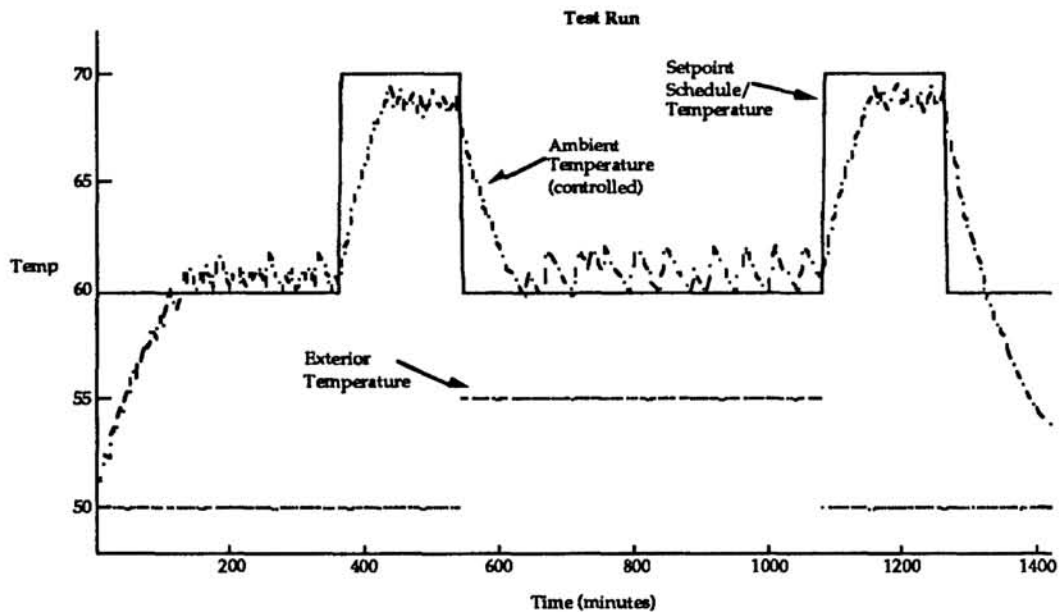

Figure 3: Time-Temperature Plot of Controlled Environment for a Test Run

Figure 1 shows how the learning progresses with the number of trials. Current results show that the learning of the complete schedule (of the six time-temperature pairs) requiring 288 control steps, can be accomplished in only 43 trials. (Given binary

output, the controller could have in the worst case executed $10^{86}$ ($\sim 2^{288}$) trials to learn the complete schedule.)

More details on the learning ability using the reinforcement learning strategy are available from the time-temperature plots of the trial and test runs in Figures 2 and 3. As the learning progresses to the forty-third trial, the controller learns to continuously heat up or cool down to the desired temperature (Figure 2). To further test the learning generalizations on the schedule, the trained network was tested on a different environment where the exterior temperature profile (and the therefore the heating and cooling rates) was different from the one used for training. Figure 3 shows the schedule that is maintained. Because the controller encounters different cooling rates in the test run, some learning still occurs as evident form Figure 3. However, all six setpoints were reached in the proper sequence. In essence, this test shows that the controller has generalized on the heating and cooling control law, independent of the setpoints and the heating and cooling rates.

## 5 CONCLUSIONS

We have developed a new learning strategy based on reinforcement learning that can be used to learn setpoint schedules for continuous processes. The experimental results have demonstrated good learning performance. However, a number of interesting extensions to this work are possible. For instance, the handicapped mode exploration of control can be better controlled for faster learning, if more information on the desired or possible trajectory is known. Another area of investigation would be the area of state encoding. In our approach, the nonlinear encoding of the system state was assumed uniform at different regions of the control space. In applications where the system with high nonlinearity, different nonlinear coding could be used adaptively to improve the state representation. Finally, other formulations of reinforcement learning algorithms, besides ASE/AHC, should also be explored. One such possibility is Watkins' Q-learning [Watkins 89].

## References

[Guha 90] A. Guha and A. Mathur, *Setpoint Control Based on Reinforcement Learning*, Proceedings of IJCNN 90, Washington D.C., January 1990.

[Barto 83] A.G. Barto, R.S. Sutton, and C.W. Anderson, *Neuronlike Adaptive Elements That Can Solve Difficult Learning Control Problems*, IEEE Transactions on Systems, Man, and Cybernetics, Vol. SMC-13, No. 5, September/October 1983.

[Michie 68] D. Michie and R. Chambers, *Machine Intelligence*, E. Dale and D. Michie (eds.), Oliver and Boyd, Edinburgh, 1968, p. 137.

[Rosen 88] B. E. Rosen, J. M. Goodwin, and J. J. Vidal, *Learning by State Recurrence Detection*, IEEE Conference on Neural Information Processing Systems - Natural and Synthetic, AIP Press, 1988.

[Watkins 89] C.J.C.H. Watkins, *Learning from Delayed Rewards*, Ph. D. Dissertation, King's College, May 1989.
